# A PROGRAMMABLE ANALOG NEURAL COMPUTER AND SIMULATOR

Paul Mueller*, Jan Van der Spiegel, David Blackman*, Timothy Chiu, Thomas Clare, Joseph Dao, Christopher Donham, Tzu-pu Hsieh, Marc Loinaz
*Dept.of Biochem. Biophys., Dept. of Electrical Engineering.
University of Pennsylvania, Philadelphia Pa.

## ABSTRACT

This report describes the design of a programmable general purpose analog neural computer and simulator. It is intended primarily for real-world real-time computations such as analysis of visual or acoustical patterns, robotics and the development of special purpose neural nets. The machine is scalable and composed of interconnected modules containing arrays of neurons, modifiable synapses and switches. It runs entirely in analog mode but connection architecture, synaptic gains and time constants as well as neuron parameters are set digitally. Each neuron has a limited number of inputs and can be connected to any but not all other neurons. For the determination of synaptic gains and the implementation of learning algorithms the neuron outputs are multiplexed, A/D converted and stored in digital memory. Even at moderate size of $10^3$ to $10^5$ neurons computational speed is expected to exceed that of any current digital computer.

## OVERVIEW

The machine described in this paper is intended to serve as a general purpose programmable neuron analog computer and simulator. Its architecture is loosely based on the cerebral cortex in the sense that there are separate neurons, axons and synapses and that each neuron can receive only a limited number of inputs. However, in contrast to the biological system, the connections can be modified by external control permitting exploration of different architectures in addition to adjustment of synaptic weights and neuron parameters.

The general architecture of the computer is shown in Fig. 1. The machine contains large numbers of the following separate elements: **neurons, synapses, routing switches** and **connection lines**. Arrays of these elements are fabricated on VLSI chips which are mounted on planar chip carriers each of which forms a separate module. These modules are connected directly to neighboring modules. Neuron arrays are arranged in rows and columns and are surrounded by synaptic and axon arrays.

The machine runs entirely in analog mode. However, connection architectures, synaptic gains and neuron parameters such as thresholds and time constants are set by a digital computer. For determining synaptic weights in a learning mode, time segments of the outputs from neurons are multiplexed, digitized and stored in digital memory.

The modular design allows expansion to any degree and at moderate to large size, i.e. $10^3$ to $10^5$ neurons, operational speed would exceed that of any currently available digital computer.

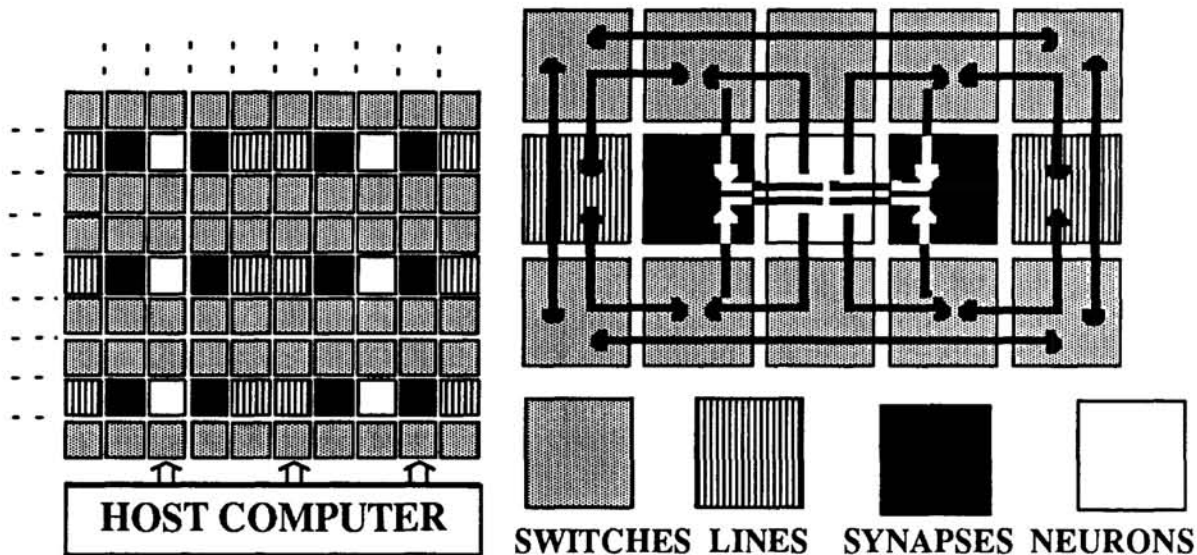

HOST COMPUTER     SWITCHES  LINES   SYNAPSES NEURONS

**Figure 1.** Layout and general architecture. The machine is composed of different modules shown here as squares. Each module contains on a VLSI chip an array of components (neurons, synapses or switches) and their control circuits. Our prototype design calls for 50 neuron modules for a total of 800 neurons each having 64 synapses.

The insert shows the direction of data flow through the modules. Outputs from each neuron leave north and south and are routed through the switch modules east and west and into the synapse modules from north and south. They can also bypass the synapse modules north and south. Input to the neurons through the synapses is from east and west. Power and digital control lines run north and south.

## THE NEURON MODULES

Each neuron chip contains 16 neurons, an analog multiplexer and control logic. (See Figs. 2 & 3.)

Input-output relations of the neurons are idealized versions of a typical biological neuron. Each unit has an adjustable threshold (bias), an adjustable minimum output value at threshold and a maximum output (See Fig. 4). Output time constants are selected on the switch chips. The neuron is based on an earlier design which used discrete components (Mueller and Lazzaro, 1986).

Inputs to each neuron come from synapse chips east and west (SIR, SIL), outputs (NO) go to switch chips north and south. Each neuron has a second input that sets the minimum output at threshold which is common for all neurons on the chip and selected through a separate synapse line. The threshold is set from one of the synapses connected to a fixed voltage. An analog multiplexer provides neuron output to a common line, OM, which connects to an A/D converter.

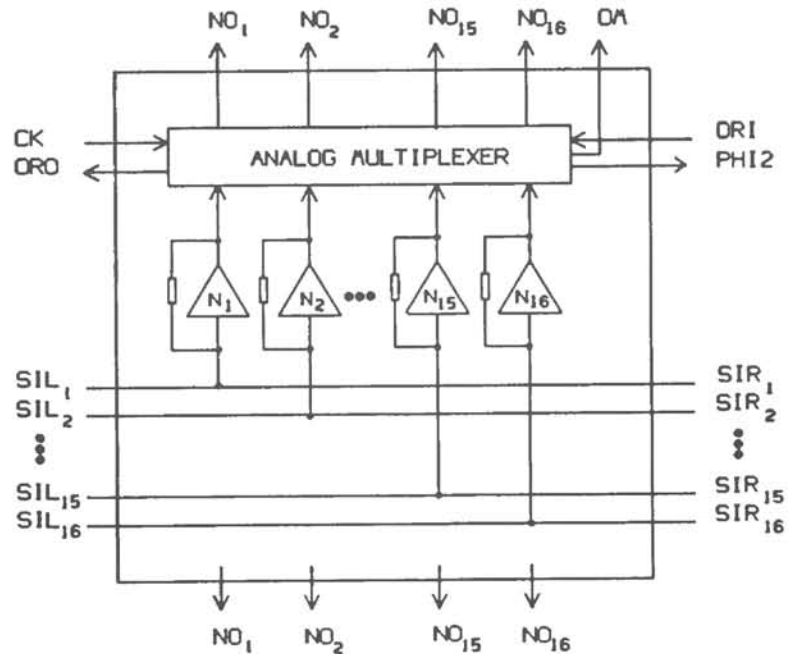

**Figure 2.** Block diagram of the neuron chip containing 16 neurons.

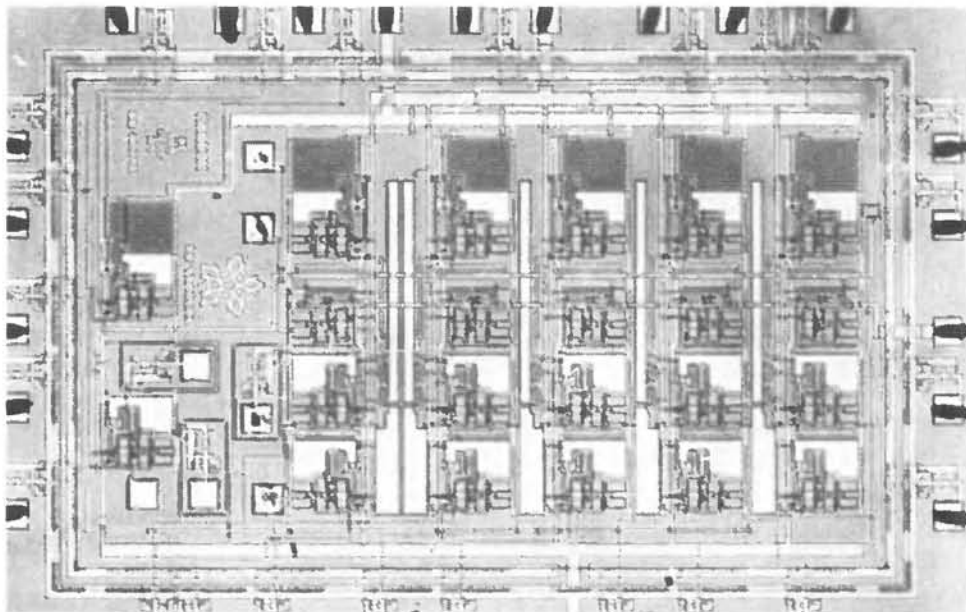

**Figure 3.** Photograph of a test chip containing 5 neurons. A more recent version has only one output sign.

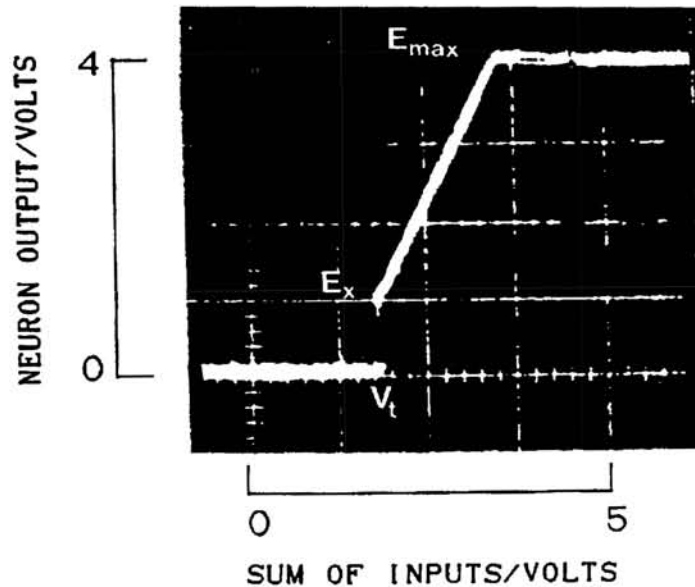

**Figure 4.** Transfer characteristic obtained from a neuron on the chip shown in Fig.3. Each unit has an adjustable threshold, $V_t$ which was set here to 1.5V, a linear transfer region above threshold, an adjustable minimum output at threshold $E_x$ set to 1V and a maximum output, $E_{max}$.

## THE SYNAPSE MODULES

Each synapse chip contains a 32 * 16 array of synapses. The synaptic gain of each synapse is set by serial input from the computer and is stored at each synapse. Dynamic range of the synapse gains covers the range from 0 to 10 with 5 bit resolution, a sixth bit determines the sign. The gains are implemented by current mirrors which scale the neuron output after it has been converted from a voltage to a current.

The modifiable synapse designs reported in the literature use either analog or digital signals to set the gains (Schwartz, et. al., 1989, Raffel, et.al, 1987, Alspector and Allen, 1987). We chose the latter method because of its greater reproducibility and because direct analog setting of the gains from the neuron outputs would require a prior knowledge of and commitment to a particular learning algorithm. Layout and performance of the synapse module are shown in Figs. 5-7. As seen in Fig. 7a, the synaptic transfer function is linear from 0 to 4 V.

The use of current mirrors permits arbitrary scaling of the synaptic gains (weights) with trade off between range and resolution limited to 5 bits. Our current design calls for a minimum gain of 1/32 and a maximum of 10. The lower end of the dynamic range is determined by the number of possible inputs per neuron which when active should not drive the neuron output to its limit, whereas the high gain values are needed in situations where a single or very few synapses must be effective such as in the copying of activity from one neuron to another or for veto inhibition. The digital nature of the synaptic gain control does not allow straight forward implementation of a logarithmic gain scale. Fig. 7b. shows two possible relations between digital code and synaptic gain. In one case the total gain is the sum of 5 individual gains each controlled by one bit. This leads inevitably to jumps in the gain curve. In a second case a linear 3 bit gain is multiplied by four different constants controlled by the 4th and 5th bit. This scheme affords a better approximation to a logarithmic scale. So far we have implemented only the first scheme.

Although the resolution of an individual synapse is limited to 5 bits, several synapses driven by one neuron can be combined through switching, permitting greater resolution and dynamic range.

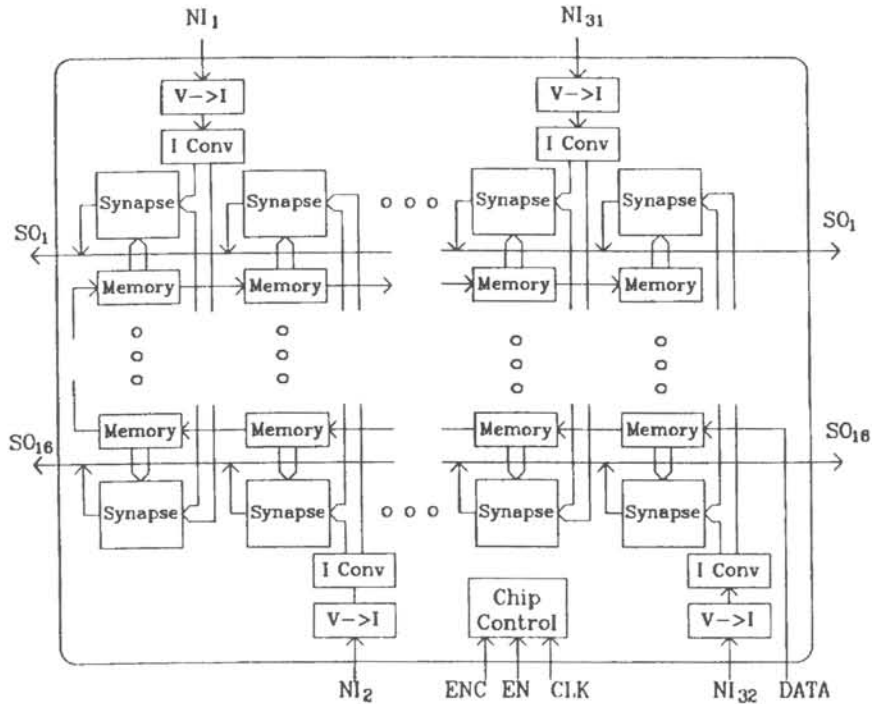

**Figure 5.** Diagram of the synapse module. Each synapse gain is set by a 5 bit word stored in local memory. The memory is implemented as a quasi dynamic shift register that reads the gain data during the programming phase. Voltage to current converters transform the neuron output (NI) into a current. I Conv are current mirrors that scale the currents with 5 bit resolution. The weighted currents are summed on a common line to the neuron input (SO).

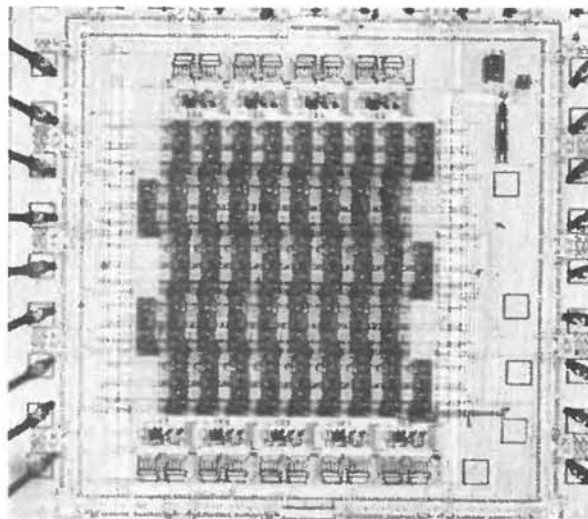

**Figure 6.** Photograph of a synapse test chip.

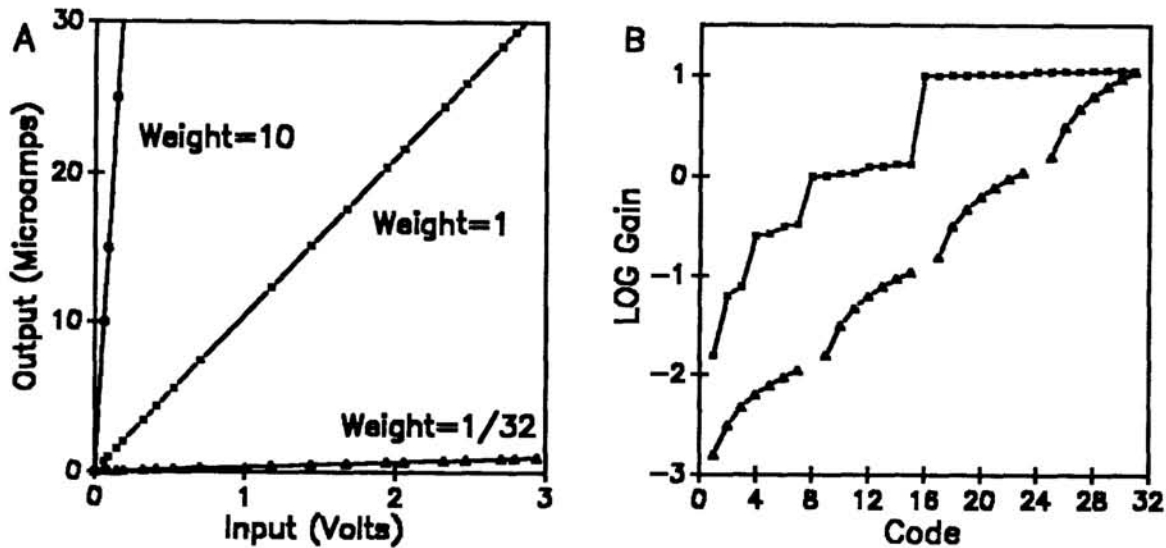

**Figure 7a.** Synapse transfer characteristics for three different settings. The data were obtained from the chip shown in Fig. 6. **b.** Digital code vs. synaptic gain, squares are current design, triangles use a two bit exponent.

## THE SWITCH MODULES

The switch modules serve to route the signals between neurons and allow changes to the connection architecture. Each module contains a 32*32 cross point array of analog switches which are set by serial digital input. There is also a set of serial switches that can disconnect input and output lines. In addition to switches the modules contain circuits which control the time constants of the synapse transfer function (see Figs. 8 & 9). The switch performance is summarized in Table 1.

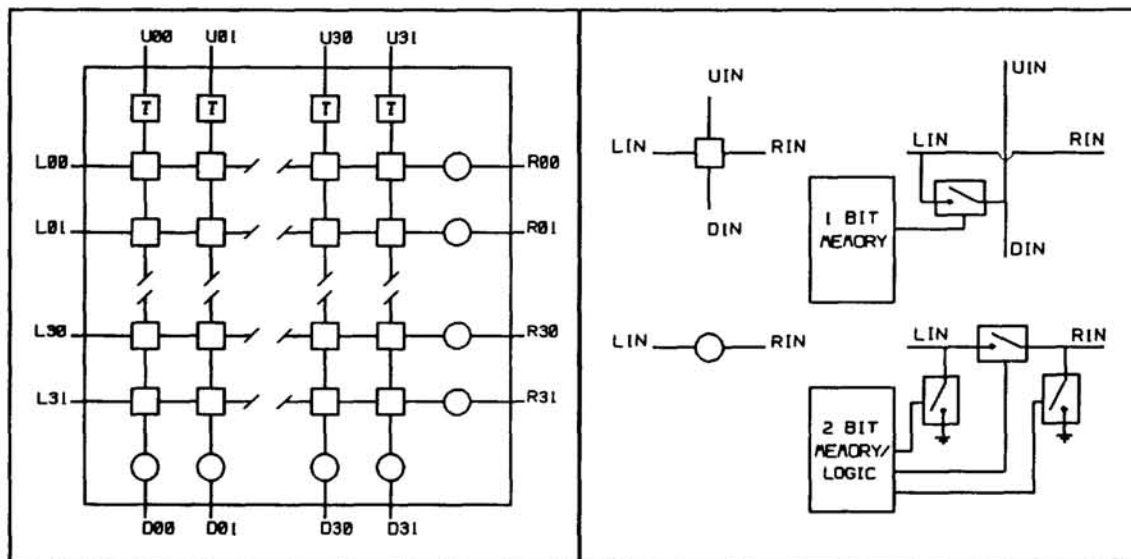

**Figure 8.** Diagram of switching fabric. Squares and circles represent switch cells which connect the horizontal and vertical connectors or cut the conductors. The units labeled T represent adjustable time constants.

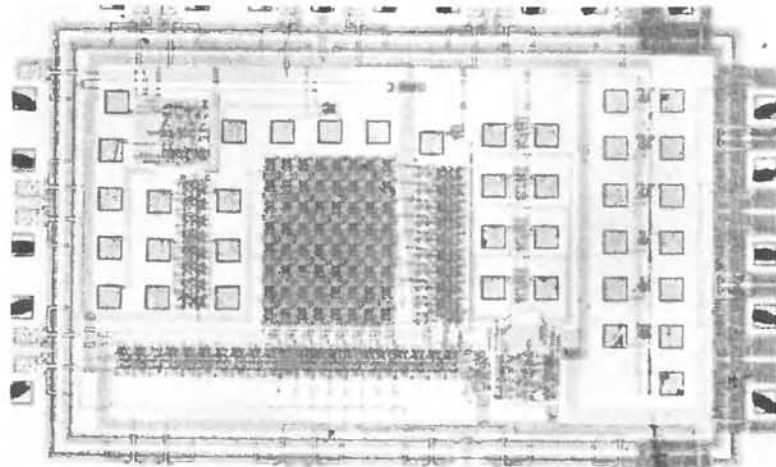

**Figure 9.** Photograph of a switch module test chip.

**TABLE 1.** Switch Chip Performance

| Process | 3u CMOS | Input capacitance | < 1pF |
|---|---|---|---|
| On resistance | < 3 KOhm | Array download time | 2us |
| Off resistance | > 1 TOhm | Memory/switch size | 75u x 90u |

## ADJUSTMENT OF SYNAPTIC TIME CONSTANTS

For the analysis or generation of temporal patterns as they occur in motion or speech, adjustable time constants of synaptic transfer must be available (Mueller, 1988). Low pass filtering of the input signal to the synapse with 4 bit control of the time constant over a range of 5 to 500 ms is sufficient to deal with real world data. By combining the low passed input with a direct input of opposite sign, both originating from the same neuron, the typical "ON" and "OFF" responses which serve as measures of time after beginning and end of events and are common in biological systems can be obtained.

Several designs are being considered for implementing the variable low pass filter. Since not all synapses need to have this feature, the circuit will be placed on only a limited number of lines on the switch chip.

## PACKAGING

All chips are mounted on identical quad surface mount carriers. Input and output lines are arranged at right angles with identical leads on opposite sides. The chip carriers are mounted on boards.

## SOFTWARE CONTROL AND OPERATION

Connections, synaptic gains and time constants are set from the central computer either manually or from libraries containing connection architectures for specific tasks. Eventually we envision developing a macro language that would generate subsystems and link them into a larger architecture. Examples are feature specific receptor fields, temporal pattern analyzers, or circuits for motion control. The connection routing is done under graphic control or through routing routines as they are used in circuit board design.

The primary areas of application include real-world real-time or compressed time pattern analysis, robotics, the design of dedicated neural circuits and the exploration of different learning algorithms. Input to the machine can come from sensory transducer arrays such as an electronic retina, cochlea (Mead, 1989) or tactile sensors. For other computational tasks, input is provided by the central digital computer through activation of selected neuron populations via threshold control. It might seem that the limited number of inputs per neuron restricts the computations performed by any one neuron. However the results obtained by one neuron can be copied through a unity gain synapse to another neuron which receives the appropriate additional inputs.

In performance mode the machine could exceed by orders of magnitude the computational speed of any currently available digital computer. A rough estimate of attainable speed can be made as follows: A network with $10^3$ neurons each receiving 100 inputs with synaptic transfer time constants ranging from 1 ms to 1 s, can be described by $10^3$ simultaneous differential equations. Assuming an average step length of 10 us and 10 iterations per step, real time numerical solutions of this system on a digital machine would require approximately $10^{11}$ FLOPS. Microsecond time constants and the computation of threshold non-linearities would require a computational speed equivalent to $>10^{12}$ FLOPS on a digital computer and this seems a reasonable estimate of the computational power of our machine. Furthermore, in contrast to digital multiprocessors, computational power would scale linearly with the number of neurons and connections.

## Acknowledgements

Supported by grants from ONR (N00014-89-J-1249) and NSF (EET 166685).

## References

Alspector, J., Allen, R.B. A neuromorphic VLSI learning system. Advanced research in VLSI. *Proceedings of the 1987 Stanford Conference,* (1987).

Mead, C. Analog VLSI and Neural Systems. Addison Wesley, Reading, Ma (1989).

Mueller, P. Computation of temporal Pattern Primitives in a Neural Net for Speech Recognition. *International Neural Network Society.* First Annual Meeting, Boston Ma., (1988).

Mueller, P., Lazzaro, J. A Machine for Neural Computation of Acoustical Patterns. *AIP Conference Proceedings,* 151:321-326, (1986).

Raffel, J.I., Mann, J.R., Berger, R., Soares, A.M., Gilbert, S., A Generic Architecture for Wafer-Scale neuromorphic Systems. *IEEE First International Conference on Neural Networks,* San Diego, CA. (1987).

Schwartz, D., Howard, R., Hubbard, W., A Programmable Analog Neural Network Chip, *J. of Solid State Circuits,* (to be published).